# Comparison Training for a Rescheduling Problem in Neural Networks

**Didier Keymeulen**
Artificial Intelligence Laboratory
Vrije Universiteit Brussel
Pleinlaan 2, 1050 Brussels
Belgium

**Martine de Gerlache**
Prog Laboratory
Vrije Universiteit Brussel
Pleinlaan 2, 1050 Brussels
Belgium

## Abstract

Airline companies usually schedule their flights and crews well in advance to optimize their crew pools activities. Many events such as flight delays or the absence of a member require the crew pool rescheduling team to change the initial schedule (rescheduling). In this paper, we show that the neural network comparison paradigm applied to the backgammon game by Tesauro (Tesauro and Sejnowski, 1989) can also be applied to the rescheduling problem of an aircrew pool. Indeed both problems correspond to choosing the best solution from a set of possible ones without ranking them (called here best choice problem). The paper explains from a mathematical point of view the architecture and the learning strategy of the backpropagation neural network used for the best choice problem. We also show how the learning phase of the network can be accelerated. Finally we apply the neural network model to some real rescheduling problems for the Belgian Airline (Sabena).

## 1 Introduction

Due to merges, reorganizations and the need for cost reduction, airline companies need to improve the efficiency of their manpower by optimizing the activities of their crew pools as much as possible. A standard scheduling of flights and crews is usually made well in advance but many events, such as flight delays or the absence of a crew member make many schedule changes (rescheduling) necessary.

Each day, the CPR[1] team of an airline company has to deal with these perturbations. The problem is to provide the best answer to these regularly occurring perturbations and to limit their impact on the general schedule. Its solution is hard to find and usually the CPR team calls on full reserve crews. An efficient rescheduling tool taking into account the experiences of the CPR team could substantially reduce the costs involved in rescheduling notably by limiting the use of a reserve crew.

The paper is organized as follow. In the second section we describe the rescheduling task. In the third section we argue for the use of a neural network for the rescheduling task and we apply an adequate architecture for such a network. Finally in the last section, we present results of experiments with schedules based on actual schedules used by Sabena.

## 2    Rescheduling for an Airline Crew Pool

When a pilot is unavailable for a flight it becomes necessary to replace him, e.g. to reschedule the crew. The rescheduling starts from a list of potential substitute pilots (PSP) given by a scheduling program based generally on operation research or expert system technology (Steels, 1990). The PSP list obtained respects legislation and security rules fixing for example the number of flying hours per month, the maximum number of consecutive working hour and the number of training hours per year and their schedule. From the PSP list, the CPR team selects the best candidates taking into account the schedule stability and equity. The *schedule stability* requires that possible perturbations of the schedule can be dealt with with only a minimal rescheduling effort. This criterion ensures work stability to the crew members and has an important influence on their social behavior. The *schedule equity* ensures the equal distribution of the work and payment among the crew members during the schedule period.

One may think to solve this rescheduling problem in the same way as the scheduling problem itself using software tools based on operational research or expert system approach. But this is inefficient for two reasons, first, the scheduling issued from a scheduling system and its adaptation to obtain an acceptable schedule takes days. Second this system does not take into account the previous schedule. It follows that the updated one may differ significantly from the previous one after each perturbation. This is unacceptable from a pilot's point of view. Hence a specific procedure for rescheduling is necessary.

## 3    Neural Network Approach

The problem of reassigning a new crew member to replace a missing member can be seen as the problem of finding the best pilot in a pool of potential substitute pilots (PSP), called the *best choice* problem.

To solve the best choice problem, we choose the neural network approach for two reasons. First the rules used by the expert are not well defined: to find the best PSP,

the expert associates implicitly a score value to each profile. The learning approach is precisely well suited to integrate, in a short period of time, the expert knowledge given in an implicit form. Second, the neural network approach was applied with success to board-games e.g. the Backgammon game described by Tesauro (Tesauro and Sejnowski, 1989) and the Nine Men's Morris game described by Braun (Braun and al., 1991). These two games are also examples of best choice problem where the player chooses the best move from a set of possible ones.

### 3.1 Profile of a Potential Substitute Pilot

To be able to use the neural network approach we have to identify the main features of the potential substitute pilot and to codify them in terms of rating values (de Gerlache and Keymeulen, 1993). We based our coding scheme on the way the expert solves a rescheduling problem. He identifies the relevant parameters associated with the PSP and the perturbed schedule. These parameters give three types of information. A first type describes the previous, present and future occupation of the PSP. The second type represents information not in the schedule such as the human relationship factors. The associated values of these two types of parameters differ for each PSP. The last type of parameters describes the context of the rescheduling, namely the characteristics of the schedule. This last type of parameters are the same for all the PSP. All these parameters form the profile of a PSP associated to a perturbed schedule. At each rescheduling problem corresponds one perturbed schedule $j$ and a group of $n$ $PSP^i$ to which we associate a $Profile^i_j \equiv (PSP^i, Perturbed\_Schedule_j)$. Implicitly, the expert associates a rating value between 0 and 1 to each parameter of the $Profile^i_j$ based on respectively its little or important impact on the resulting schedule if the $PSP^i$ was chosen. The rating value reflects the relative importance of the parameters on the stability and the equity of the resulting schedule obtained after the pilots substitution.

### 3.2 Dual Neural Network

It would have been possible to get more information from the expert than only the best profile. One of the possibilities is to ask him to score every profile associated with a perturbed planning. From this association we could immediately construct a *scoring function* which couples each profile with a specific value, namely its score. Another possibility is to ask the expert to rank all profiles associated with a perturbed schedule. The corresponding *ranking function* couples each profile with a value such that the values associated with the profiles of the same perturbed schedule order the profiles according to their rank. The decision making process used by the rescheduler team for the aircrew rescheduling problem does not consist in the evaluation of a scoring or ranking function. Indeed only the knowledge of the best profile is useful for the rescheduling process.

From a neural network architectural point of view, because the ranking problem is a generalization of the best choice problem, a same neural network architecture can be used. But the difference between the best choice problem and the scoring problem is such that two different neural network architectures are associated to them. As we show in this section, although a backpropagation network is sufficient to learn a scoring function, its architecture, its learning and its retrieval procedures must be

adapted to learn the best profile. Through a mathematical formulation of the best
choice problem, we show that the comparison paradigm of Tesauro (Tesauro, 1989)
is suited to the best choice problem and we suggest how to improve the learning
convergence.

### 3.2.1   Comparing Function

For the best choice problem the expert gives the best profile $Profile_j^{Best}$ associated
with the perturbed schedule $j$ and that for $m$ perturbed schedules. The problem
consists then to learn the mapping of the $m * n$ profiles associated with the $m$
perturbed schedules into the $m$ best profiles, one for each perturbed schedule. One
way to represent this association is through a *comparing function*. This function
has as input a profile, represented by a vector $\vec{X}_j^i$, and returns a single value. When
a set of profiles associated with a perturbed schedule are evaluated by the function,
it returns the lowest value for the best profile. This comparing function integrates
the information given by the expert and is sufficient to reschedule any perturbed
schedule solved in the past by the expert. Formally it is defined by:

$$Compare_j^i = \mathcal{C}(Profile_j^i) \qquad (1)$$

$$Compare_j^{Best} < Compare_j^i \quad \begin{cases} \forall j & \text{with} \quad j = 1, \ldots, m \\ \forall i \neq Best & \text{with} \quad i = 1, \ldots, n \end{cases}$$

The value of $Compare_j^i$ are not known a priori and have only a meaning when they
are compared to the value $Compare_j^{Best}$ of the comparing function for the best
profile.

### 3.2.2   Geometrical Interpretation

To illustrate the difference between the neural network learning of a scoring function
and a comparing function, we propose a geometrical interpretation in the case of
a linear network having as input vectors (profiles) $\vec{X}_j^1, \ldots, \vec{X}_j^i, \ldots, \vec{X}_j^n$ associated
with a perturbed schedule $j$.

The learning of a scoring function which associates a score $Score_j^i$ with each input
vector $\vec{X}_j^i$ consists in finding a hyperplane in the input vector space which is tangent
to the circles of center $\vec{X}_i^j$ and radius $Score_i^j$ (Fig. 1). On the contrary the learning
of a comparing function consists to obtain the equation of an hyperplane such that
the end-point of the vector $\vec{X}_j^{Best}$ is nearer the hyperplane than the end-points of
the other input vectors $\vec{X}_j^i$ associated with the same perturbed schedule $j$ (Fig. 1).

### 3.2.3   Learning

We use a neural network approach to build the comparing function and the mean
squared error as a measure of the quality of the approximation. The comparing
function is approximated by a non-linear function: $\mathcal{C}(Profile_j^i) = \mathcal{NL}(\vec{W}, \vec{X}_j^i)$
where $\vec{W}$ is the weight vector of the neural network (e.g backpropagation network).
The problem of finding $\mathcal{C}$ which has the property of (1) is equivalent to finding the
function $\mathcal{C}$ that minimizes the following error function (Braun and al., 1991) where
$\Phi$ is the *sigmoid* function :

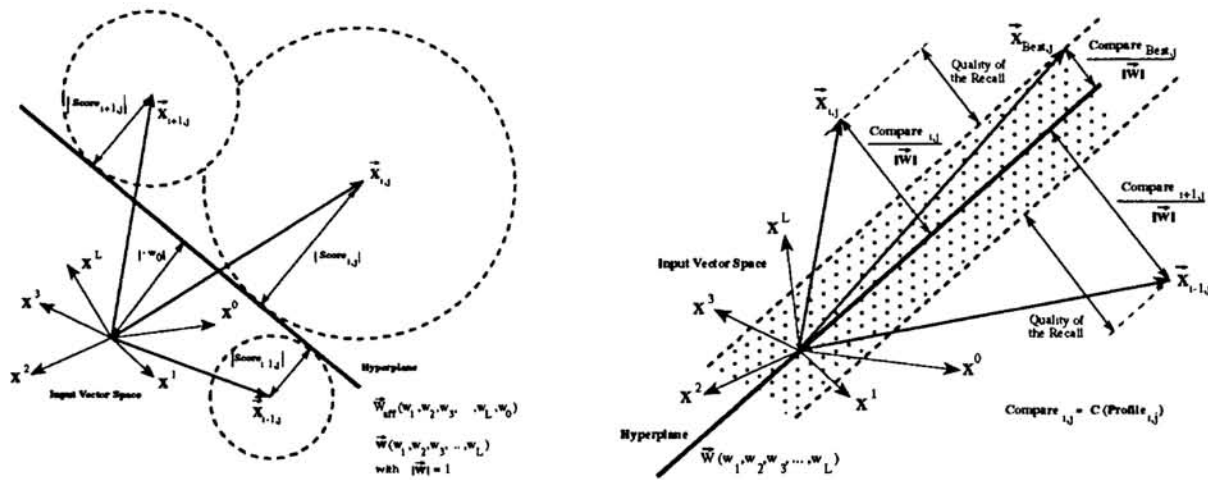

Figure 1: Geometrical Interpretation of the learning of a Scoring Function (Rigth) and a Comparing Function (Left)

$$\mathcal{E}_\Phi = \left(\frac{1}{m*n}\right) \sum_{j=1}^m \sum_{\substack{i=1 \\ i \neq Best}}^n \left(\Phi\left[\mathcal{C}(Profile_j^{Best}) - \mathcal{C}(Profile_j^i)\right]\right)^2 \qquad (2)$$

To obtain the weight vector which minimizes the error function (2), we use the property that the $-\vec{grad}\mathcal{E}_\Phi(\vec{W})$ points in the direction in which the error function will decrease at the fastest possible rate. To update the weight we have thus to calculate the partial derivative of (2) with each components of the weight vector $\vec{W}$: it is made of a product of three factors. The evaluation of the first two factors (the sigmoid and the derivative of the sigmoid) is immediate. The third factor is the partial derivative of the non-linear function $\mathcal{NL}$ which is generally calculated by using the *generalized delta rule* learning law (Rumelhart and McClelland, 1986).

Unlike the linear associator network, for the backpropagation network, the error function (2) is not equivalent to the error function where the difference $\vec{X}_j^{Best} - \vec{X}_j^i$ is associated with the input vector of the backpropagation network because:

$$\mathcal{NL}(\vec{W}, \vec{X}_j^{Best}) - \mathcal{NL}(\vec{W}, \vec{X}_j^i) \neq \mathcal{NL}(\vec{W}, \vec{X}_j^{Best} - \vec{X}_j^i) \qquad (3)$$

By consequence to calculate the three factors of the partial derivative of (2), we have to introduce separately at the bottom of the network the input vector of the best profile $\vec{X}_j^{Best}$ and the input vector of a less good profile $\vec{X}_j^i$. Then we have to memorize their partial contribution at each node of the network and multiply their contributions before updating the weight. Using this way to evaluate the derivative of (2) and to update the weight, the simplicity of the generalized delta rule learning law has disappeared.

### 3.2.4  Architecture

Tesauro (Tesauro and Sejnowski, 1989) proposes an architecture, that we call *dual neural network*, and a learning procedure such that the simplicity of the generalized delta rule learning law can still be used (Fig. 2). The same kind of architecture, called *siamese network*, was recently used by Bromley for the signature verification (Bromley and al., 1994). The dual neural network architecture and the learning strategy are justified mathematically at one hand by the decomposition of the partial derivative of the error function (2) in a sum of two terms and at the other hand by the asymmetry property of the sigmoid and its derivative.

The architecture of the dual neural network consists to duplicate the multi-layer network approximating the comparing function (1) and to connect the output of both to a unique output node through a positive unit weight for the left network and negative unit weight for the right network. During the learning a couple of profiles is presented to the dual neural network: a best profile $\vec{X}_j^{Best}$ and a less good profile $\vec{X}_j^i$. The desired value at the output node of the dual neural network is 0 when the left network has for input the best profile and the right network has for input a less good profile and 1 when these profiles are permuted. During the recall we work only with one of the two multi-layer networks, suppose the left one (the choice is of no importance because they are exactly the same). The profiles $\vec{X}_j^i$ associated with a perturbed schedule $j$ are presented at the input of the left multi-layer network. The best profile is the one having the lowest value at the output of the left multi-layer network.

Through this mathematical formulation we can use the suggestion of Braun to improve the learning convergence (Braun and al., 1991). They propose to replace the positive and negative unit weight between the output node of the multi-layer networks and the output node of the dual neural network by respectively a weight value equal to $\mathcal{V}$ for the left network and $-\mathcal{V}$ for the right network. They modify the value of $\mathcal{V}$ by applying the generalized delta rule which has no significant impact on the learning convergence. By manually increasing the factor $\mathcal{V}$ during the learning procedure, we improve considerably the learning convergence due to its asymmetric impact on the derivative of $\mathcal{E}_\Phi(\vec{W})$ with $\vec{W}$: the modification of the weight vector is greater for couples not yet learned than for couples already learned.

## 4  Results

The experiments show the ability of our model to help the CPR team of the Sabena Belgian Airline company to choose the best profile in a group of PSPs based on the learned expertise of the team. To codify the profile we identify 15 relevant parameters. They constitute the input of our neural network. The training data set was obtained by analyzing the CPR team at work during 15 days from which we retain our training and test perturbed schedules.

We consider that the network has learned when the comparing value of the best profile is less than the comparing value of the other profiles and that for all training perturbed schedules. At that time $\mathcal{E}_\Phi(\vec{W})$ is less than .5 for every couple of profiles. The left graph of Figure 3 shows the evolution of the mean error over the couples

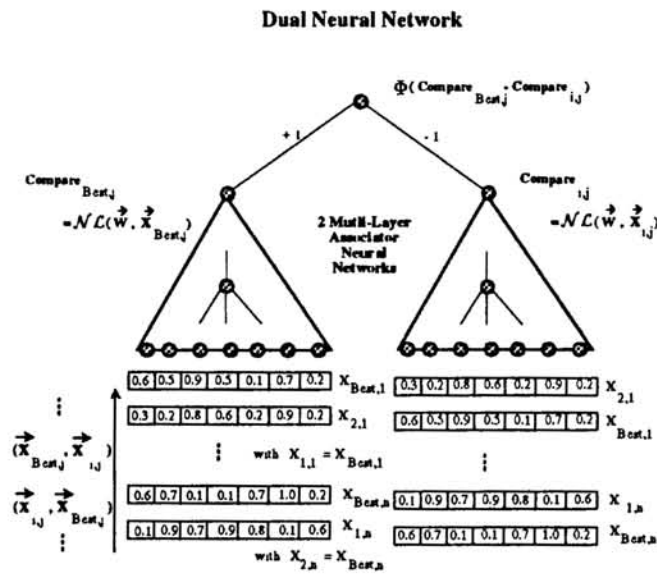

Figure 2: The training of a dual neural network.

during the training. The right graph shows the improvement of the convergence when the weight $\mathcal{V}$ is increased regularly during the training process.

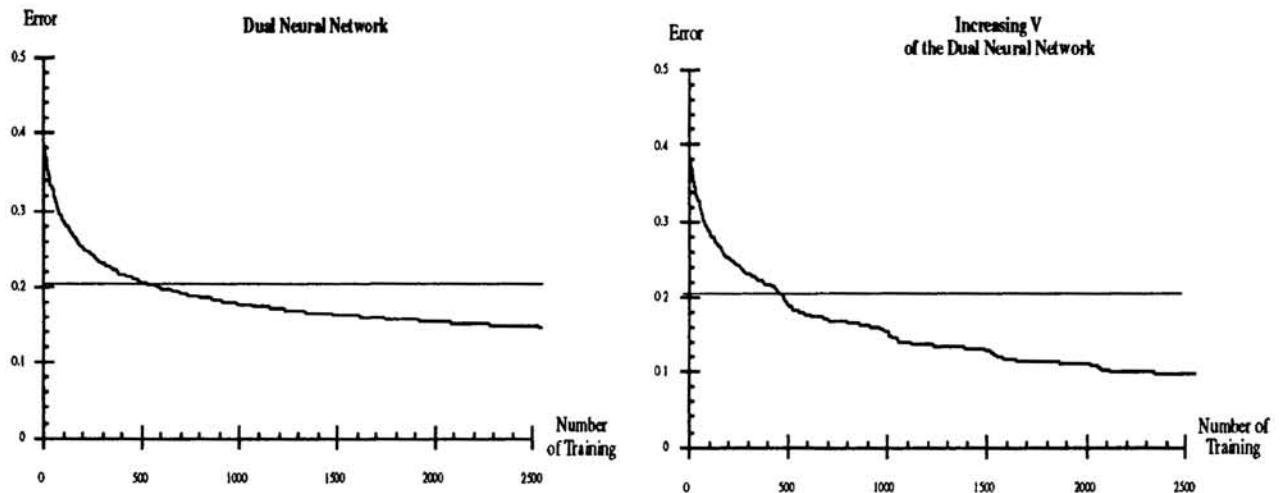

Figure 3: Convergence of the dual neural network architecture.

The network does not converge when we introduce contradictory decisions in our training set. It is possible to resolve them by adding new context parameters in the coding scheme of the profile.

After learning, our network shows generalization capacity by retrieving the best profile for a new perturbed schedule that is similar to one which has already been learned. The degree of similarity required for the generalization remains a topic for further study.

## 5   Conclusion

In conclusion, we have shown that the rescheduling problem of an airline crew pool can be stated as a decision making problem, namely the identification of the best potential substitute pilot. We have stressed the importance of the codification of the information used by the expert to evaluate the best candidate. We have applied the neural network learning approach to help the rescheduler team in the rescheduling process by using the experience of already solved rescheduling problems. By a mathematical analysis we have proven the efficiency of the dual neural network architecture. The mathematical analysis permits also to improve the convergence of the network. Finally we have illustrated the method on rescheduling problems for the Sabena Belgian Airline company.

**Acknowledgments**

We thank the Scheduling and Rescheduling team of Mr. Verworst at Sabena for their valuable information given all along this study; Professors Steels and D'Hondt from the VUB and Professors Pastijn, Leysen and Declerck from the Military Royal Academy who supported this research; Mr. Horner and Mr. Pau from the Digital Europe organization for their funding. We specially thank Mr. Decuyper and Mr. de Gerlache for their advices and attentive reading.

**References**

H. Braun, J. Faulner & V. Uilrich. (1991) Learning strategies for solving the problem of planning using backpropagation. In *Proceedings of Fourth International Conference on Neural Networks and their Applications*, 671-685. Nimes, France.

J. Bromley, I. Guyon, Y. Lecun, E. Sackinger, R. Shah. (1994). Signature verification using a siamese delay neural network. In J. Cowan, G. Tesauro & J. Alspector (eds.), *Advances in Neural Information Processing Systems 1*. San Mateo, CA: Morgan Kaufmann.

M. de Gerlache & D. Keymeulen. (1993) A neural network learning strategy adapted for a rescheduling problem. In *Proceedings of Fourth International Conference on Neural Networks and their Applications*, 33-42. Nimes, France.

D. Rumelhart & J. McClelland. (1986) *Parallel Distributed Processing: Explorations in the Microstructure of Cognition I & II*. Cambridge, MA: MIT Press.

L. Steels. (1990) Components of expertise. *AI Magazine*, **11**(2):29-49.

G. Tesauro. (1989) Connectionist learning of expert preferences by comparison training. In D. S. Touretzky (ed.), *Advances in Neural Information Processing Systems 1*, 99-106. San Mateo, CA: Morgan Kaufmann.

G. Tesauro & T.J. Sejnowski. (1989) A parallel network that learns to play backgammon. *Artificial Intelligence*, **39**:357-390.

## Footnotes

[1]Crew Pool Rescheduler
